# On Efficient Heuristic Ranking of Hypotheses

**Steve Chien, Andre Stechert, and Darren Mutz**
Jet Propulsion Laboratory, California Institute of Technology
4800 Oak Grove Drive, M/S 525-3660, Pasadena, CA 91109-8099
steve.chien@jpl.nasa.gov, Voice: (818) 306-6144 FAX: (818) 306-6912
Content Areas: Applications (Stochastic Optimization),Model Selection Algorithms

## Abstract

This paper considers the problem of learning the ranking of a set of alternatives based upon incomplete information (e.g., a limited number of observations). We describe two algorithms for hypothesis ranking and their application for probably approximately correct (PAC) and expected loss (EL) learning criteria. Empirical results are provided to demonstrate the effectiveness of these ranking procedures on both synthetic datasets and real-world data from a spacecraft design optimization problem.

## 1 INTRODUCTION

In many learning applications, the cost of information can be quite high, imposing a requirement that the learning algorithms glean as much usable information as possible with a minimum of data. For example:

- In speedup learning, the expense of processing each training example can be significant [Tadepalli92].
- In decision tree learning, the cost of using all available training examples when evaluating potential attributes for partitioning can be computationally expensive [Musick93].
- In evaluating medical treatment policies, additional training examples imply suboptimal treatment of human subjects.
- In data-poor applications, training data may be very scarce and learning as well as possible from limited data may be key.

This paper provides a statistical decision-theoretic framework for the ranking of parametric distributions. This framework will provide the answers to a wide range of questions about algorithms such as: how much information is enough? At what point do we have adequate information to rank the alternatives with some requested confidence?

The remainder of this paper is structured as follows. First, we describe the hypothesis ranking problem more formally, including definitions for the probably approximately correct (PAC) and expected loss (EL) decision criteria. We then define two algorithms for establishing these criteria for the hypothesis ranking problem – a recursive hypothesis selection algorithm and an adjacency based algorithm. Next, we describe empirical tests demonstrating the effectiveness of these algorithms as well as documenting their improved performance over a standard algorithm from the statistical ranking literature. Finally, we describe related work and future extensions to the algorithms.

## 2   HYPOTHESIS RANKING PROBLEMS

Hypothesis ranking problems, an extension of hypothesis selection problems, are an abstract class of learning problems where an algorithm is given a set of hypotheses to rank according to *expected utility* over some unknown distribution, where the expected utility must be estimated from training data.

In many of these applications, a system chooses a single alternative and never revisits the decision. However, some systems require the ability to investigate several options (either serially or in parallel), such as in beam search or iterative broadening, where the ranking formulation is most appropriate. Also, as is the case with evolutionary approaches, a system may need to populate future alternative hypotheses on the basis of the ranking of the current population[Goldberg89].

In any hypothesis evaluation problem, always achieving a correct ranking is impossible in practice, because the actual underlying probability distributions are unavailable and there is always a (perhaps vanishingly) small chance that the algorithms will be unlucky because only a finite number of samples can be taken. Consequently, rather than always requiring an algorithm to output a correct ranking, we impose probabilistic criteria on the rankings to be produced. While several families of such requirements exist, in this paper we examine two, the *probably approximately correct* (PAC) requirement from the computational learning theory community [Valiant84] and the *expected loss* (EL) requirement frequently used in decision theory and gaming problems [Russell92].

The expected utility of a hypothesis can be estimated by observing its values over a finite set of training examples. However, to satisfy the PAC and EL requirements, an algorithm must also be able to reason about the potential difference between the estimated and true utilities of each hypotheses. Let $U_i$ be the true expected utility of hypothesis $i$ and let $\hat{U}_i$ be the estimated expected utility of hypothesis $i$. Without loss of generality, let us presume that the proposed ranking of hypotheses is $U_1 > U_2 >, ..., > U_{k-1} > U_k$. The PAC requirement states that for some user-specified $\epsilon$ with probability $1 - \delta$:

$$\bigwedge_{i=1}^{k-1} [(U_i + \epsilon) > MAX(U_{i+1}, ..., U_k)] \tag{1}$$

Correspondingly, let the loss $L$ of selecting a hypothesis $H_1$ to be the best from a set of $k$ hypotheses $H_1, ..., H_k$ be as follows.

$$L(H_1, \{H_1, ..., H_k\}) = MAX(0, MAX(U_2, ..., U_k) - U_1) \tag{2}$$

and let the loss $RL$ of a ranking $H_1, ..., H_k$ be as follows.

$$RL(H_1, ..., H_k) = \sum_{i=1}^{k-1} L(H_i, \{H_{i+1}, ..., H_k\}) \tag{3}$$

A hypothesis ranking algorithm which obeys the expected loss requirement must produce rankings that on average have less than the requested expected loss bound.

Consider ranking the hypotheses with expected utilities: $U_1 = 1.0, U_2 = 0.95, U_3 = 0.86$. The ranking $U_2 > U_1 > U_3$ is a valid PAC ranking for $\epsilon = 0.06$ but not for $\epsilon = 0.01$ and has an observed loss of $0.05 + 0 = 0.05$.

However, while the confidence in a pairwise comparison between two hypotheses is well understood, it is less clear how to ensure that desired confidence is met in the set of comparisons required for a selection or the more complex set of comparisons required for a ranking. Equation 4 defines the confidence that $U_i + \epsilon > U_j$, when the distribution underlying the utilities is normally distributed with unknown and unequal variances.

$$\gamma = \phi \left( (\hat{U}_{i-j} + \epsilon) \frac{\sqrt{n}}{\hat{S}_{i-j}} \right) \tag{4}$$

where $\phi$ represents the cumulative standard normal distribution function, and $n$, $\hat{U}_{i-j}$, and $\hat{S}_{i-j}$ are the size, sample mean, and sample standard deviation of the blocked differential distribution, respectively[1].

Likewise, computation of the expected loss for asserting an ordering between a pair of hypotheses is well understood, but the estimation of expected loss for an entire ranking is less clear. Equation 5 defines the expected loss for drawing the conclusion $U_i > U_j$, again under the assumption of normality (see [Chien95] for further details).

$$EL[U_i > U_j] = \frac{\hat{S}_{i-j} e^{-0.5n(\frac{\hat{U}_{i-j}}{\hat{S}_{i-j}})^2}}{\sqrt{2\pi n}} + \frac{\hat{U}_{i-j}}{\sqrt{2\pi}} \int_{-\frac{\hat{U}_{i-j}\sqrt{n}}{\hat{S}_{i-j}}}^{\infty} e^{-0.5z^2} dz \tag{5}$$

In the next two subsections, we describe two interpretations for estimating the likelihood that an overall ranking satisfies the PAC or EL requirements by estimating and combining pairwise PAC errors or EL estimates. Each of these interpretations lends itself directly to an algorithmic implementation as described below.

## 2.1   RANKING AS RECURSIVE SELECTION

One way to determine a ranking $H_1, ..., H_k$ is to view ranking as recursive selection from the set of remaining candidate hypotheses. In this view, the overall ranking error, as specified by the desired confidence in PAC algorithms and the loss threshold in EL algorithms, is first distributed among $k - 1$ *selection errors* which are then further subdivided into *pairwise comparison errors*. Data is then sampled until the estimates of the pairwise comparison error (as dictated by equation 4 or 5) satisfy the bounds set by the algorithm.

Thus, another degree of freedom in the design of recursive ranking algorithms is the method by which the overall ranking error is ultimately distributed among individual pairwise comparisons between hypotheses. Two factors influence the way in which we compute error distribution. First, our model of error combination determines how the error allocated for individual comparisons or selections combines into overall ranking error and thus how many candidates are available as targets for the distribution. Using Bonferroni's inequality, one combine errors additively, but a more conservative approach might be to assert that because the predicted "best" hypothesis may change during sampling in the worst case the conclusion might depend on all possible pairwise comparisons and thus the error should be distributed among all $\binom{n}{2}$ pairs of hypotheses[2]).

Second, our policy with respect to allocation of error among the candidate comparisons or selections determines how samples will be distributed. For example, in some contexts, the consequences of early selections far outweigh those of later selections. For these scenarios, we have implemented ranking algorithms that divide overall ranking error unequally in favor of earlier selections[3]. Also, it is possible to divide selection error into pairwise error unequally based on estimates of hypothesis parameters in order to reduce sampling cost (for example, [Gratch94] allocates error rationally).

Within the scope of this paper, we only consider algorithms that: (1) combine pairwise error into selection error additively, (2) combine selection error into overall ranking error additively and (3) allocate error equally at each level.

One disadvantage of recursive selection is that once a hypothesis has been selected, it is removed from the pool of candidate hypotheses. This causes problems in rare instances when, while sampling to increase the confidence of some later selection, the estimate for a hypothesis' mean changes enough that some previously selected hypothesis no longer dominates it. In this case, the algorithm is restarted taking into account the data sampled so far.

These assumptions result in the following formulations (where $\delta(U_1 \rhd_\epsilon \{U_2, ..., U_k\})$ is used to denote the error due to the action of selecting hypothesis 1 under Equation 1 from the set $\{H_1, ..., H_k\}$ and $\delta(U_1 \rhd \{U_2, ..., U_k\})$ denotes the error due to selection loss in situations where Equation 2 applies):

$$
\begin{aligned}
\delta_{rec}(U_1 > U_2 > ... > U_k) = \quad &\delta_{rec}(U_2 > U_3 > ... > U_k) \\
&+\delta(U_1 \rhd_\epsilon \{U_2, ..., U_k\})
\end{aligned}
\tag{6}
$$

where $\delta_{rec}(U_k) = 0$ (the base case for the recursion) and the selection error is as defined in [Chien95]:

$$
\delta(U_1 \rhd_\epsilon \{U_2, ..., U_k\}) = \sum_{i=2}^{k} \delta_{1,i}
\tag{7}
$$

using Equation 4 to compute pairwise confidence.

Algorithmically, we implement this by:

1. sampling a default number of times to seed the estimates for each hypothesis mean and variance,
2. allocating the error to selection and pairwise comparisons as indicated above,
3. sampling until the desired confidences for successive selections is met, and
4. restarting the algorithm if any of the hypotheses means changed significantly enough to change the overall ranking.

An analogous recursive selection algorithm based on expected loss is defined as follows.

$$
\begin{aligned}
EL_{rec}(U_1 > U_2 > ... > U_k) = \quad &EL_{rec}(U_2 > U_3 > ... > U_k) \\
&+EL(U_1 \rhd \{U_2, ..., U_k\})
\end{aligned}
\tag{8}
$$

where $EL_{rec}(U_k) = 0$ and the selection EL is as defined in [Chien95]:

$$
EL(U_1 \rhd \{U_2, ..., U_k\}) = \sum_{i=2}^{k} EL(U_1, U_i)
\tag{9}
$$

## 2.2   RANKING BY COMPARISON OF ADJACENT ELEMENTS

Another interpretation of ranking confidence (or loss) is that only adjacent elements in the ranking need be compared. In this case, the overall ranking error is divided directly into $k-1$ pairwise comparison errors. This leads to the following confidence equation for the PAC criteria:

$$\delta_{adj}(U_1 > U_2 > ... > U_k) = \sum_{i=1}^{k-1} \delta_{i,i+1} \qquad (10)$$

And the following equation for the EL criteria.

$$EL_{adj}(U_1 > U_2 > ... > U_k) = \sum_{i=1}^{k-1} EL(U_i, U_{i+1}) \qquad (11)$$

Because ranking by comparison of adjacent hypotheses does not establish the dominance between non-adjacent hypotheses (where the hypotheses are ordered by observed mean utility), it has the advantage of requiring fewer comparisons than recursive selection (and thus may require fewer samples than recursive selection). However, for the same reason, adjacency algorithms may be less likely to correctly bound probability of correct selection (or average loss) than the recursive selection algorithms. In the case of the PAC algorithms, this is because $\epsilon$-dominance is not necessarily transitive. In the case of the EL algorithms, it is because expected loss is not additive when considering two hypothesis relations sharing a common hypothesis. For instance, the size of the blocked differential distribution may be different for each of the pairs of hypotheses being compared.

## 2.3   OTHER RELEVANT APPROACHES

Most standard statistical ranking/selection approaches make strong assumptions about the form of the problem (e.g., the variances associated with underlying utility distribution of the hypotheses might be assumed known and equal). Among these, Turnbull and Weiss [Turnbull84] is most comparable to our PAC-based approach[4]. Turnbull and Weiss treat hypotheses as normal random variables with unknown mean and unknown and unequal variance. However, they make the additional stipulation that hypotheses are independent. So, while it is still reasonable to use this approach when the candidate hypotheses are not independent, excessive statistical error or unnecessarily large training set sizes may result.

## 3   EMPIRICAL PERFORMANCE EVALUATION

We now turn to empirical evaluation of the hypothesis ranking techniques on real-world datasets. This evaluation serves three purposes. First, it demonstrates that the techniques perform as predicted (in terms of bounding the probability of incorrect selection or expected loss). Second, it validates the performance of the techniques as compared to standard algorithms from the statistical literature. Third, the evaluation demonstrates the robustness of the new approaches to real-world hypothesis ranking problems.

An experimental trial consists of solving a hypothesis ranking problem with a given technique and a given set of problem and control parameters. We measure performance by (1) how well the algorithms satisfy their respective criteria; and (2) the number of samples taken. Since the performance of these statistical algorithms on any single trial provides little information about their overall behavior, each trial is repeated multiple times and the results are averaged across 100 trials. Because

Table 1: Estimated expected total number of observations to rank DS-2 spacecraft designs. Achieved probability of correct ranking is shown in parenthesis.

| k | $\gamma^*$ | $\frac{a}{c}$ | TURNBULL | $PAC_{rec}$ | $PAC_{adj}$ |
|---|---|---|---|---|---|
| 10 | 0.75 | 2 | 534 (0.96) | 144 (1.00) | 92 (0.98) |
| 10 | 0.90 | 2 | 667 (0.98) | 160 (1.00) | 98 (1.00) |
| 10 | 0.95 | 2 | 793 (0.99) | 177 (1.00) | 103 (0.99) |

Table 2: Estimated expected total number of observations and expected loss of an incorrect ranking of DS-2 penetrator designs.

| Parameters | | $EL_{rec}$ | | $EL_{adj}$ | |
|---|---|---|---|---|---|
| k | $H^*$ | Samples | Loss | Samples | Loss |
| 10 | 0.10 | 152 | 0.005 | 77 | 0.014 |
| 10 | 0.05 | 200 | 0.003 | 90 | 0.006 |
| 10 | 0.02 | 378 | 0.003 | 139 | 0.003 |

the PAC and expected loss criteria are not directly comparable, the approaches are analyzed separately.

Experimental results from synthetic datasets are reported in [Chien97]. The evaluation of our approach on artificially generated data is used to show that: (1) the techniques correctly bound probability of incorrect ranking and expected loss as predicted when the underlying assumptions are valid even when the underlying utility distributions are inherently hard to rank, and (2) that the PAC techniques compare favorably to the algorithm of Turnbull and Weiss in a wide variety of problem configurations.

The test of real-world applicability is based on data drawn from an actual NASA spacecraft design optimization application. This data provides a strong test of the applicability of the techniques in that all of the statistical techniques make some form of normality assumption - yet the data in this application is highly non-normal.

Tables 1 and 2 show the results of ranking 10 penetrator designs using the PAC-based, Turnbull, and expected loss algorithms In this problem the utility function is the depth of penetration of the penetrator, with those cases in which the penetrator does not penetrate being assigned zero utility. As shown in Table 1, both PAC algorithms significantly outperformed the Turnbull algorithm, which is to be expected because the hypotheses are somewhat correlated (via impact orientations and soil densities). Table 2 shows that the $EL_{rec}$ expected loss algorithm effectively bounded actual loss but the $EL_{adj}$ algorithm was inconsistent.

## 4 DISCUSSION AND CONCLUSIONS

There are a number of areas of related work. First, there has been considerable analysis of hypothesis selection problems. Selection problems have been formalized using a Bayesian framework [Moore94, Rivest88] that does not require an initial sample, but uses a rigorous encoding of prior knowledge. Howard [Howard70] also details a Bayesian framework for analyzing learning cost for selection problems. If one uses a hypothesis selection framework for ranking, allocation of pairwise errors can be performed rationally [Gratch94]. Reinforcement learning work [Kaelbling93] with immediate feedback can also be viewed as a hypothesis selection problem.

In summary, this paper has described the hypothesis ranking problem, an extension to the hypothesis selection problem. We defined the application of two decision criteria, *probably approximately correct* and *expected loss*, to this problem. We then defined two families of algorithms, recursive selection and adjacency, for solution of hypothesis ranking problems. Finally, we demonstrated the effectiveness of these algorithms on both synthetic and real-world datasets, documenting improved performance over existing statistical approaches.

## Footnotes

[1]Note that in our approach we *block* examples to further reduce sampling complexity. Blocking forms estimates by using the difference in utility between competing hypotheses on each observed example. Blocking can significantly reduce the variance in the data when the hypotheses are not independent. It is trivial to modify the formulas to address the cases in which it is not possible to block data (see [Moore94, Chien95] for further details).

[2]For a discussion of this issue, see pp. 18-20 of [Gratch93].

[3]Space constraints preclude their description here.

[4]PAC-based approaches have been investigated extensively in the statistical ranking and selection literature under the topic of *confidence interval based* algorithms (see [Haseeb85] for a review of the recent literature).

# References

[Bechhofer54] R.E. Bechhofer, "A Single-sample Multiple Decision Procedure for Ranking Means of Normal Populations with Known Variances," *Annals of Math. Statistics* (25) 1, 1954 pp. 16-39.

[Chien95] S. A. Chien, J. M. Gratch and M. C. Burl, "On the Efficient Allocation of Resources for Hypothesis Evaluation: A Statistical Approach," *IEEE Trans. Pattern Analysis and Machine Intelligence 17 (7)*, July 1995, pp. 652-665.

[Chien97] S. Chien, A. Stechert, and D. Mutz, "Efficiently Ranking Hypotheses in Machine Learning," JPL-D-14661, June 1997. *Available online at http://www-aig.jpl.nasa.gov/public/www/pas-bibliography.html*

[Goldberg89] D. Goldberg, Genetic Algorithms in Search, Optimization and Machine Learning, Add. Wes., 1989.

[Govind81] Z. Govindarajulu, "The Sequential Statistical Analysis," American Sciences Press, Columbus, OH, 1981.

[Gratch92] J. Gratch and G. DeJong, "COMPOSER: A Probabilistic Solution to the Utility Problem in Speed-up Learning," Proc. AAAI92, San Jose, CA, July 1992, pp. 235-240.

[Gratch93] J. Gratch, "COMPOSER: A Decision-theoretic Approach to Adaptive Problem Solving," Tech. Rep. UIUCDCS-R-93-1806, Dept. Comp. Sci., Univ. Illinois, May 1993.

[Gratch94] J. Gratch, S. Chien, and G. DeJong, "Improving Learning Performance Through Rational Resource Allocation," Proc. AAAI94, Seattle, WA, August 1994, pp. 576-582.

[Greiner92] R. Greiner and I. Jurisica, "A Statistical Approach to Solving the EBL Utility Problem," Proc. AAAI92, San Jose, CA, July 1992, pp. 241-248.

[Haseeb85] R. M. Haseeb, *Modern Statistical Selection*, Columbus, OH: Am. Sciences Press, 1985.

[Hogg78] R. V. Hogg and A. T. Craig, Introduction to Mathematical Statistics, Macmillan Inc., London, 1978.

[Howard70] R. A. Howard, Decision Analysis: Perspectives on Inference, Decision, and Experimentation," Proceedings of the IEEE 58, 5 (1970), pp. 823-834.

[Kaelbling93] L. P. Kaelbling, Learning in Embedded Systems, MIT Press, Cambridge, MA, 1993.

[Minton88] S. Minton, Learning Search Control Knowledge: An Explanation-Based Approach, Kluwer Academic Publishers, Norwell, MA, 1988.

[Moore94] A. W. Moore and M. S. Lee, "Efficient Algorithms for Minimizing Cross Validation Error," Proc. ML94, New Brunswick, MA, July 1994.

[Musick93] R. Musick, J. Catlett and S. Russell, "Decision Theoretic Subsampling for Induction on Large Databases," Proc. ML93, Amhert, MA, June 1993, pp. 212-219.

[Rivest88] R. L. Rivest and R. Sloan, A New Model for Inductive Inference," Proc. 2nd Conference on Theoretical Aspects of Reasoning about Knowledge, 1988.

[Russell92] S. Russell and E. Wefald, Do the Right Thing: Studies in Limited Rationality, MIT Press, MA.

[Tadepalli92] P. Tadepalli, "A theory of unsupervised speedup learning," Proc. AAAI92,, pp. 229-234.

[Turnbull84] Turnbull and Weiss, "A class of sequential procedures for k-sample problems concerning normal means with unknown unequal variances," in *Design of Experiments: ranking and selection*, T. J. Santner and A. C. Tamhane (eds. ), Marcel Dekker, 1984.

[Valiant84] L. G. Valiant, "A Theory of the Learnable," Communications of the ACM 27, (1984), pp. 1134-1142.